# Neural Network Diagnosis of Avascular Necrosis from Magnetic Resonance Images

**Armando Manduca**
Dept. of Physiology and Biophysics
Mayo Clinic
Rochester, MN 55905

**Paul Christy**
Dept. of Diagnostic Radiology
Mayo Clinic
Rochester, MN 55905

**Richard Ehman**
Dept. of Diagnostic Radiology
Mayo Clinic
Rochester, MN 55905

## Abstract

Avascular necrosis (AVN) of the femoral head is a common yet potentially serious disorder which can be detected in its very early stages with magnetic resonance imaging. We have developed multi-layer perceptron networks, trained with conjugate gradient optimization, which diagnose AVN from single magnetic resonance images of the femoral head with 100% accuracy on training data and 97% accuracy on test data.

## 1 INTRODUCTION

Diagnostic radiology may be a very natural field of application for neural networks, since a simple answer is desired from a complex image, and the learning process that human experts undergo is to a large extent a supervised learning experience based on looking at large numbers of images with known interpretations. Although many workers have applied neural nets to various types of 1-dimensional medical data (e.g. ECG and EEG waveforms), little work has been done on applying neural nets to diagnosis directly from medical images.

We wanted to explore the use of neural networks in diagnostic radiology by (1) starting with a simple but real diagnostic problem, and (2) using only actual data. We chose the diagnosis of avascular necrosis from magnetic resonance images as an ideal initial problem, because: the area in question is small and well-defined, its size and shape do not vary greatly between individuals, the condition (if present) is usually visible even at low spatial and gray level resolution on a single image, and real data is readily available.

Avascular necrosis (AVN) is the deterioration of tissue due to a disruption in the blood supply. AVN of the femoral head (the ball at the upper end of the femur which fits into the socket formed by the hip bone) is an increasingly common clinical problem, with potentially crippling effects. Since the sole blood supply to the femoral head in adults traverses the femoral neck, AVN often occurs following hip fracture (e.g., Bo Jackson). It is now apparent that AVN can also occur as a side effect of treatment with corticosteroid drugs, which are commonly used for immunosuppression in transplant patients as well as for patients with asthma, rheumatoid arthritis and other autoimmune diseases. Although the pathogenesis of AVN secondary to corticosteroid use is not well understood, 6 - 10% of such patients appear to develop the disorder (Ternoven et al., 1990). AVN may be detected with magnetic resonance imaging (MRI) even in its very early stages, as a low signal region within the femoral head due to loss of water-containing bone marrow. MRI is expected to play an important future role in screening patients undergoing corticosteroid therapy for AVN.

## 2   METHODOLOGY

The data set selected for analysis consisted of 125 sagittal images of femoral heads from T1-weighted MRI scans of 40 adult patients, with 51% showing evidence of AVN, from early stages to quite severe (see Fig. 1). Often both femoral heads from the same patient were selected (typically only one has AVN if the cause is fracture-related while both sometimes have AVN if the cause is secondary to drug use), and often two or three different cross-sectional slices of the same femoral head were included (the appearance of AVN can change dramatically as one steps through different cross-sectional slices). The images were digitized and 128x128 regions centered on and just containing the femoral heads were manually selected. These 128x128 subimages with 256 gray levels were averaged down to 32x32 resolution and to 16 gray levels for most of the trials (see Fig. 2).

The neural networks used to analyze the data were standard feed-forward, fully-connected multilayer perceptrons with a single hidden layer of 4 to 30 nodes and 2 output nodes. The majority of the runs were with networks of 1024 input nodes, into which the 32x32 images were placed, with gray levels scaled so the input values ranged within +0.5. In other experiments with different input features the number of input nodes varied accordingly. Conjugate gradient optimization was used for training (Kramer and Sangiovanni-Vincentelli, 1989; Barnard and Cole 1989). Training was stopped at a maximum of 50 passes through the training set, though usually convergence was achieved before this point. Each training run took less than 1 minute on a SPARCstation 2.

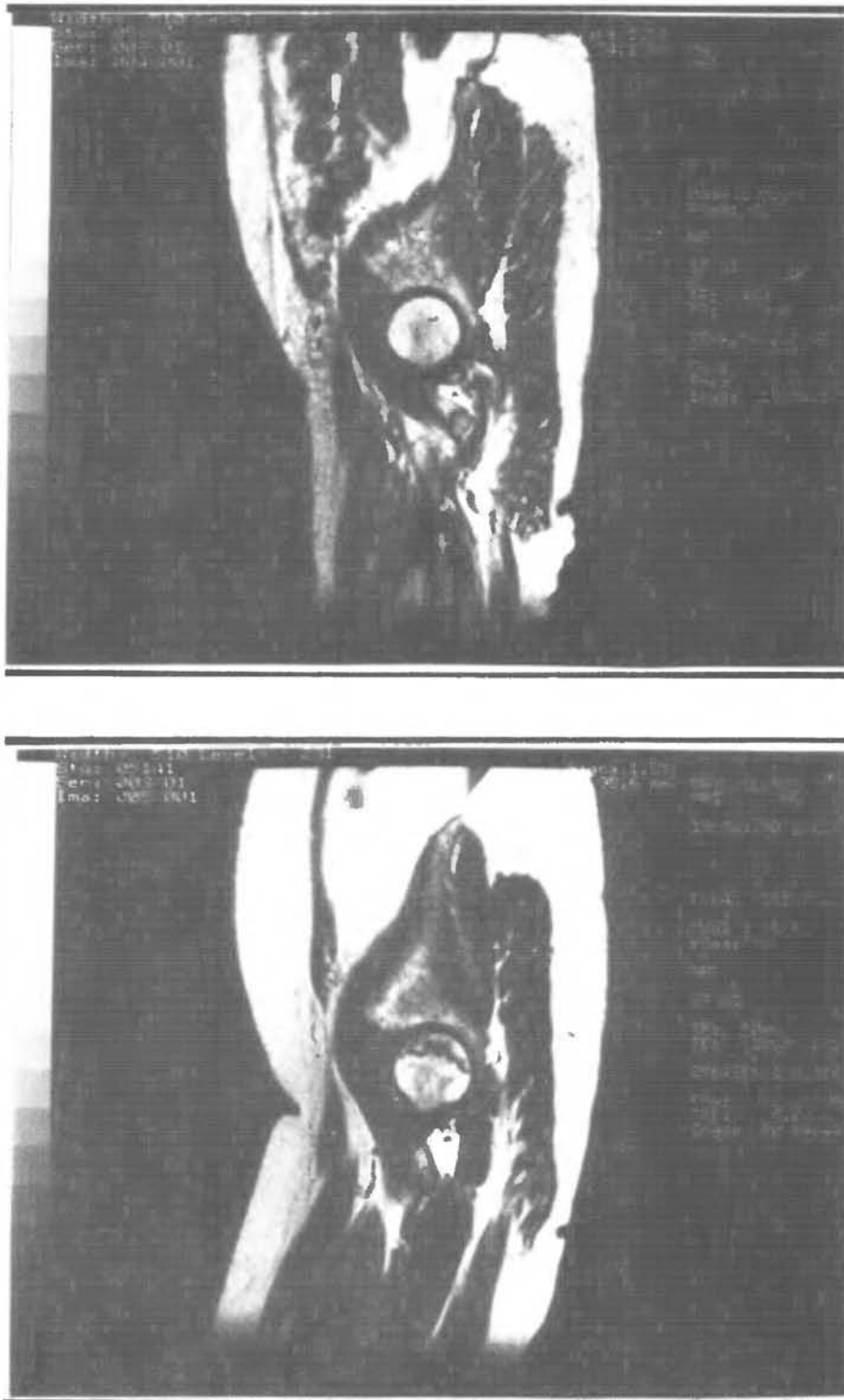

Figure 1: Representative sagittal hip T1 weighted MR images. The small circular area in the center of each picture is the femoral head (the ball joint at the upper end of the femur). The top image shows a normal femoral head; the bottom is a femoral head with severe avascular necrosis.

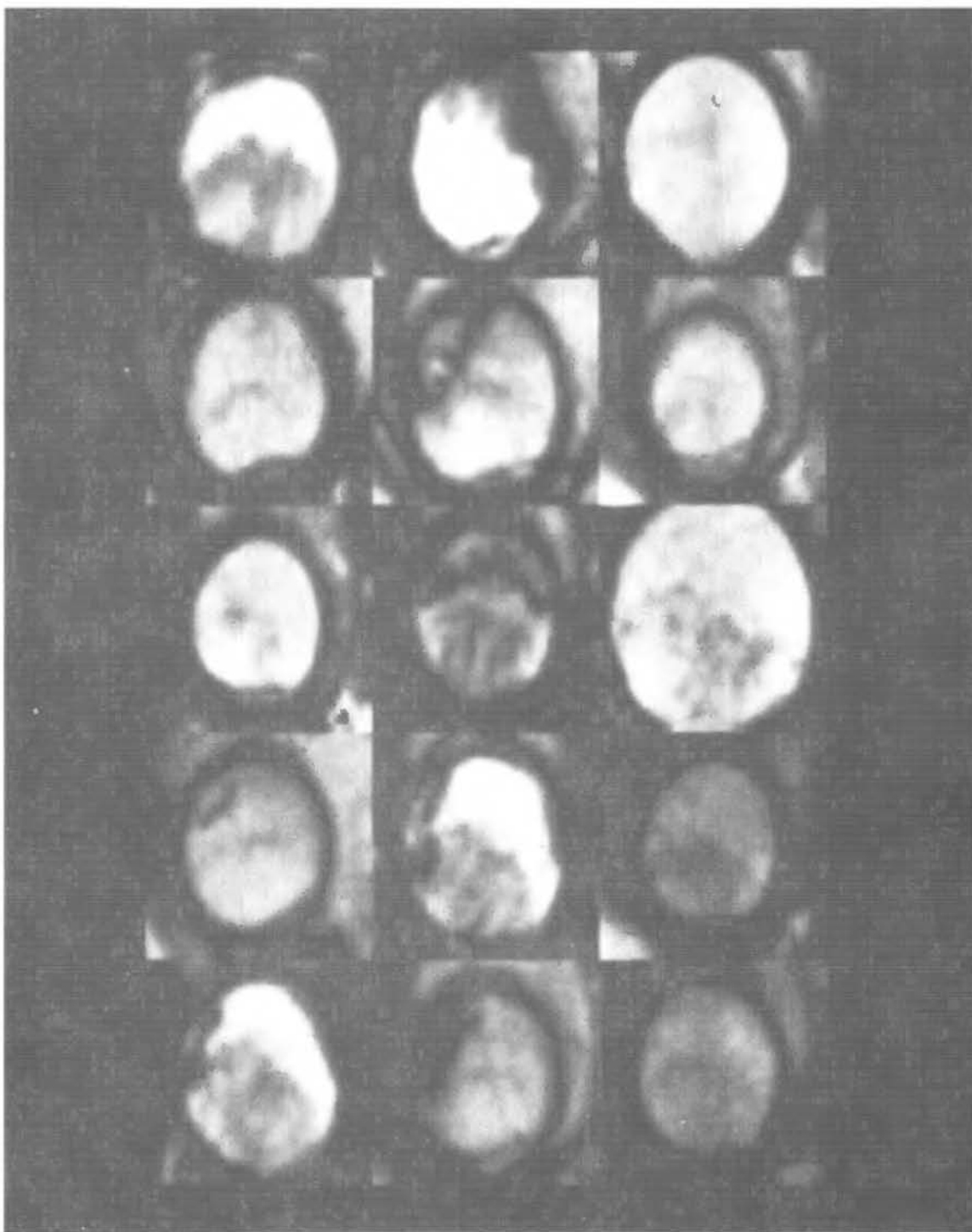

Figure 2: Sample images from our 32x32 pixel, 16 gray level data set. The five
femoral heads in the right column are free of AVN, the five in the middle column
have varying degrees of AVN, while the left column shows five images that were
particularly difficult for both the networks and untrained humans to distinguish
(only the last two have AVN).

Table 1: Diagnostic Accuracies on Test Data

(averages over 24 and 100 runs respectively)

| hidden nodes | 50% training | 80% training |
|---|---|---|
| none | 91.6% | 92.6% |
| 4 | 92.6% | 95.5% |
| 5 | 93.2% | 96.4% |
| 6 | 93.8% | 96.4% |
| 7 | 93.2% | 97.0% |
| 8 | 92.4% | 96.8% |
| 10 | 92.4% | 96.1% |
| 30 | 91.2% | 94.1% |

## 3   RESULTS

Two sets of runs with the image data were made, with the data randomly split 50%-50% and 80%-20% into training and test data sets respectively. In the first set, 4 different random splits of the data, with either half in turn serving as training or test data, and 3 different random weight initializations each were used for a total of 24 distinct runs for each network configuration. For the other set, since there was less test data, 10 different splits of the data with 10 different weight initializations each were used for a total of 100 distinct runs for each network configuration. The results are shown in Table 1. In all cases, the sensitivity and specificity were approximately equal. Standard deviations of the averages shown were typically 4.0% for the 24 run values and 3.0% for the 100 run values.

The overall data set is linearly separable, and networks with no hidden nodes readily achieved 100% on training data and better than 91% on test data. Networks with 2 or 3 hidden nodes were unable to converge on the training data much of the time, but with 4 hidden nodes convergence was restored and accuracy on test data was improved over the linear case. This accuracy increased up to 6 or 7 hidden nodes, and then began a gradual decrease as still more hidden nodes were added. This may be related to overfitting of the training data with the extra degrees of freedom, leading to poorer generalization. Adding a second hidden layer also decreased generalization accuracy.

Many other experiments were performed, using as inputs respectively: the 2-D FFT of the images, the power spectrum, features extracted with a ring-wedge detector in frequency space, the image data combined with each of the above, and multiple slight translations of the training and/or test data. None of these yielded an improvement in accuracy over the above, and no approach to date with significantly fewer than 1024 inputs maintained the high accuracies above. We are continuing experiments on other forms of reducing the dimensionality of the input data. A few experiments have been run with much larger networks, maintaining the full 128x128 resolution and 256 gray levels, but this also yields no improvement in the results.

## 4   DISCUSSION

The networks' performance at the 50% training level was comparable to that of humans with no training in radiology, who, supplied with the correct diagnosis for half of the images, averaged 92.5% accuracy on the remaining half. When the networks were trained on a larger set of data, their accuracy improved, to as high as 97.0% when 80% of the data was used for training. We expect this performance to continue to improve as larger data sets are collected.

It is difficult to compare the networks' performance to trained radiologists, who can diagnose AVN with essentially 100% accuracy, but who look at multiple cross-sectional images of far higher quality than our low-resolution, 16 gray-level data set. When presented with single images from our data set, they typically make no mistakes but set aside a few images as uncertain and strongly resist being forced to commit to an answer on those. We are currently experimenting with networks which can take inputs from multiple slices and which have an additional output representing uncertainty.

We consider the 97% accuracy achieved here to be very encouraging for further work on this problem and for the use of neural networks in more complex problems in diagnostic radiology. This is perhaps a very natural field of application for neural networks, since radiology resident training is essentially a four year experience with a very large training set, and the American College of Radiology teaching file is a classic example of a large collection of input/output training pairs (Boone et al., 1990). More complex diagnostic radiology problems may of course require fusing information from multiple images or imaging modalities, clinical data, and medical knowledge (perhaps as expert system rules). An especially intriguing possibility is that sophisticated network based systems could someday be presented with images which cannot currently be interpreted, supplied with the correct diagnosis as determined by other means, and learn to detect subtle distinctions in the images that are not apparent to human radiologists.

### References

Barnard, E. and Cole, R. (1989) "A neural-net training program based on conjugate gradient optimization", Oregon Graduate Institute, Technical report CSE 89-014.

Boone, J. M., Sigillito, V. G. and Shaber, G. S. (1990), "Neural networks in radiology: An introduction and evaluation in a signal detection task", *Medical Physics*, **17**, 234-241.

Kramer, A. and Sangiovanni-Vincentelli, A. (1989), "Efficient Parallel Learning Algorithms for Neural Networks", in D. S. Touretzky (ed.) *Advances in Neural Information Processing Systems 1*, 40-48. Morgan-Kaufmann, San Mateo, CA.

Ternoven, O. et al. (1990), "Prevalence of Asymptomatic, Clinically Occult Avascular Necrosis of the Hip in a Population at Risk", *Radiology*, **177(P)**, 104.